# Bayesian Unsupervised Learning of Higher Order Structure

Michael S. Lewicki    Terrence J. Sejnowski
lewicki@salk.edu       terry@salk.edu

The Salk Institute
Howard Hughes Medical Institute
Computational Neurobiology Lab
10010 N. Torrey Pines Rd.
La Jolla, CA 92037

## Abstract

Multilayer architectures such as those used in Bayesian belief networks and Helmholtz machines provide a powerful framework for representing and learning higher order statistical relations among inputs. Because exact probability calculations with these models are often intractable, there is much interest in finding approximate algorithms. We present an algorithm that efficiently discovers higher order structure using EM and Gibbs sampling. The model can be interpreted as a stochastic recurrent network in which ambiguity in lower-level states is resolved through feedback from higher levels. We demonstrate the performance of the algorithm on benchmark problems.

## 1  Introduction

Discovering high order structure in patterns is one of the keys to performing complex recognition and discrimination tasks. Many real world patterns have a hierarchical underlying structure in which simple features have a higher order structure among themselves. Because these relationships are often statistical in nature, it is natural to view the process of discovering such structures as a statistical inference problem in which a hierarchical model is fit to data.

Hierarchical statistical structure can be conveniently represented with Bayesian belief networks (Pearl, 1988; Lauritzen and Spiegelhalter, 1988; Neal, 1992). These

models are powerful, because they can capture complex statistical relationships among the data variables, and also mathematically convenient, because they allow efficient computation of the joint probability for any given set of model parameters. The joint probability density of a network of binary states is given by a product of conditional probabilities

$$P(S_1 \ldots S_n|\mathbf{W}) = \prod_i P(S_i|\text{pa}[S_i], \mathbf{W}) \tag{1}$$

where $\mathbf{W}$ is the weight matrix that parameterizes the model. Note that the probability of an individual state $S_i$ depends only on its parents. This probability is given by

$$P(S_i = 1|\text{pa}[S_i], \mathbf{W}) = h(\sum_j S_j w_{ji}) \tag{2}$$

where $w_{ji}$ is the weight from $S_j$ to $S_i$ ($w_{ji} = 0$ for $j < i$).

The weights specify a hierarchical prior on the input states, which are the fixed subset of states at the lowest layer of units. The active parents of state $S_i$ represent the underlying causes of that state. The function $h$ specifies how these causes are combined to give the probability of $S_i$. We assume $h$ to be the "noisy OR" function, $h(u) = 1 - \exp(-u)$, $u >= 0$.

## 2    Learning Objective

The learning objective is to adapt $\mathbf{W}$ to find the most probable explanation of the input patterns. The probability of the input data is

$$P(\mathbf{D}_{1:N}|\mathbf{W}) = \prod_n P(\mathbf{D}_n|\mathbf{W}) \tag{3}$$

$P(\mathbf{D}_n|\mathbf{W})$ is computed by marginalizing over all states of the network

$$P(\mathbf{D}_n|\mathbf{W}) = \sum_k P(\mathbf{D}_n|\mathbf{S}_k, \mathbf{W})P(\mathbf{S}_k|\mathbf{W}) \tag{4}$$

Because the number of different states, $\mathbf{S}_k$, is exponential in the number of units, computing the sum exactly is intractable and must be approximated. The nature of the learning tasks discussed here, however, allow us to make accurate approximations. A desirable property for representations is that most patterns have just one or a few possible explanations. In this case, all but a few terms $P(\mathbf{D}_n|\mathbf{S}_k, \mathbf{W})$ will be zero, and, as described below, it becomes feasible to use sampling based methods which select $\mathbf{S}_k$ according to $P(\mathbf{S}_k|\mathbf{D}_n, \mathbf{W})$.

## 3    Inferring the Internal Representation

Given the input data, finding its most likely explanation is an *inference* process. Although it is simple to calculate the probability of any particular network state, there is no simple way to determine the most probable state given input D. A general approach to this problem is Gibbs sampling (Pearl, 1988; Neal, 1992).

In Gibbs sampling, each state $S_i$ of the network is updated iteratively according to the probability of $S_i$ given the remaining states in the network. This conditional probability can be computed using

$$P(S_i|S_j : j \neq i, \mathbf{W}) \propto P(S_i|\text{pa}[S_i], \mathbf{W}) \prod_{j \in \text{ch}[S_i]} P(S_j|\text{pa}[S_j], S_i, \mathbf{W}) \qquad (5)$$

where $\text{ch}[S_i]$ indicates the children of state $S_i$. In the limit, the ensemble of states obtained by this procedure will be typical samples from $P(\mathbf{S}|\mathbf{D}, \mathbf{W})$. More generally, any subset of states can be fixed and the rest sampled.

The Gibbs equations have an interpretation in terms of a stochastic recurrent neural network in which feedback from higher levels influences the states at lower levels. For the models defined here, the probability of $S_i$ changing state given the remaining states is

$$P(S_i = 1 - S_i|S_j : j \neq i, \mathbf{W}) = \frac{1}{1 + \exp(-\Delta x_i)} \qquad (6)$$

The variable $\Delta x_i$ indicates how much changing the state $S_i$ changes the probability of the network state

$$\Delta x_i = \log h(u_i; 1 - S_i) - \log h(u_i; S_i) + \sum_{j \in \text{ch}[S_i]} \log h(u_j + \delta_{ij}; S_j) - \log h(u_j; S_j) \qquad (7)$$

where $h(u; a) = h(u)$ if $a = 1$ and $1 - h(u)$ if $a = 0$. The variable $u_i$ is the causal input to $S_i$, given by $\sum_k S_k w_{ki}$. The variable $\delta_j$ specifies the change in $u_j$ for a change in $S_i$: $\delta_{ij} = +S_j w_{ij}$ if $S_i = 0$ and $-S_j w_{ij}$ if $S_i = 1$.

The first two terms in (7) can be interpreted as the feedback from higher levels. The sum can be interpreted as the feedforward input from the children of $S_i$. Feedback allows the lower level units to use information only computable at higher levels. The feedforward terms typically dominate the expression, but the feedback becomes the determining factor when the feedforward input is ambiguous.

For general distributions, Gibbs sampling can require many samples to achieve a representative samples. But if there is little ambiguity in the internal representation, as is the goal, Gibbs sampling can be as efficient as a single feedforward pass. One potential problem is that Gibbs sampling will not work before the weights have adapted, when the representations are highly ambiguous. We show below, however, that it is not necessary to sample for long periods in order for good representations to be learned. As learning proceeds, the internal representations obtained with limited Gibbs sampling become increasingly accurate.

## 4 Adapting the Weights

The complexity of the model is controlled by placing a prior on the weights. For the form of the noisy OR function in which all weights are constrained to be positive, we assume the prior to be the product of independent gamma distributions parameterized by $\alpha$ and $\beta$. The objective function becomes

$$\mathcal{L} = P(\mathbf{D}_{1:N}|\mathbf{W})P(\mathbf{W}|\alpha, \beta) \qquad (8)$$

A simple and efficient EM-type formula for adapt the weights can be derived by setting $\partial \mathcal{L}/w_{ij}$ to zero and solving for $w_{ij}$. Using the transformations $f_{ij} = 1 -$

$\exp(-w_{ij})$ and $g_i = 1 - \exp(-u_i)$ we obtain

$$f_{ij} = \frac{\alpha - 1 + 2f_{ij} + \sum_n S_i^{(n)} S_j^{(n)} f_{ij}/g_j^{(n)}}{\alpha + \beta + \sum_n S_i^{(n)}} \qquad (9)$$

where $S^{(n)}$ is the state obtained with Gibbs sampling for the $n$th input pattern. The variable $f_{ij}$ can be interpreted as the frequency of state $S_j$ given cause $S_i$. The sum in the above expression is a weighted average of the number of times $S_j$ was active when $S_i$ was active. The ratio $f_{ij}/g_j$ weights each term in the sum inversely according to the number of different causes for $S_j$. If $S_i$ is the unique cause of $S_j$ then $f_{ij} = g_j$ and the term would have full weight.

A straightforward application of the learning algorithm would adapt all the weights at the same time. This does not produce good results, however, because there is nothing to prevent the model from learning overly strong priors. This can be prevented by adapting the weights in the upper levels after the weights in the lower levels have stabilized. This allows the higher levels to adapt to structure that is actually present in the data. We have obtained good results from both the naive method of adapting the lowest layers first and from more sophisticated methods where stability was based on how often a unit changed during the Gibbs sampling.

## 5  Examples

In the following examples, the weight prior was specified with $\alpha = 1.0$ and $\beta = 1.5$. Weights were set to random values between 0.05 and 0.15. Gibbs sampling was stopped if the maximum state change probability was less than 0.05 or after 15 sweeps through the units. Weights were reestimated after blocks of 200 patterns. Each layer of weights was adapted for 10 epochs before adapting the next layer.

**A High Order Lines Problem.** The first example illustrates that the algorithm can discover the underlying features in complicated patterns and that the higher layers can capture interesting higher order structure. The first dataset is a variant of the lines problem proposed by Földiák (1989). The patterns in the dataset are composed of horizontal and vertical lines as illustrated in figure 1. Note that, although

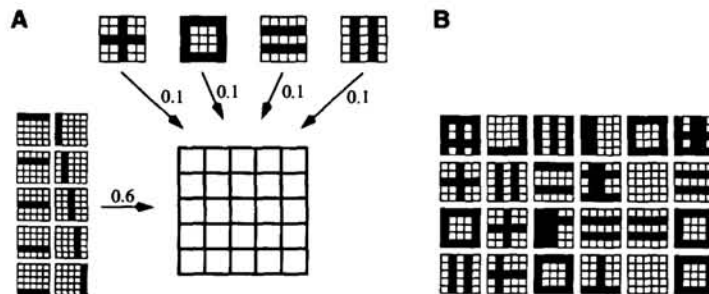

Figure 1: Dataset for the high order lines problem. (**A**) Patterns are generated by selecting one of the pattern types according to the probabilities next to the arrows. Top patterns are copied to the input. The horizontal and vertical lines on the left are selected with probability 0.3. (**B**) Typical input patterns.

the datasets are displayed on a 2-D grid, the network makes no assumptions about topography. Because the network is fully connected, all spatial arrangements of the inputs are identical. The weights learned by the network are shown in figure 2.

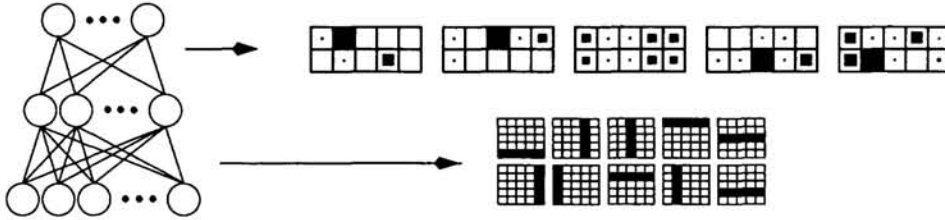

Figure 2: The weights in a 25-10-5 network after training. Blocks indicate the weights to each unit. Square size is proportional to the weight values. Second layer units capture the structure of the horizontal and vertical lines. Third layer units capture the correlations among the lines. The first unit in the third layer is active when the '||' is present. The second, fourth, and fifth units have learned to represent the '+', '=', and '□' respectively, with the remaining unit acting as a bias.

**The Shifter Problem.** The shifter problem (Hinton and Sejnowski, 1986), explained in figure 3, is important because the structure that must be discovered is in the higher order input correlations. This example also illustrates the importance of allowing high level states to influence low level states to determine the most probable internal representation. The units in the second layer can only capture second order statistics and cannot determine the direction of the shift. The only way these units can be disambiguated is to use the feedback from the units in the third layer which detect the direction of the shift by integrating the output of the units in the second layer. This allows the representation in the second layer to be "cleaned up" and makes it easier to discover the higher order structure of the global shift. The speed and reliability of the learning was tested by learning from random initial conditions. The results are shown in figure 4. Note that the best solutions have a cost of about one bit higher than the optimal cost of less than 9 bits, because top units cannot capture the fact that they are mutually exclusive.

## 6 Discussion

The methods we have described work well on these simple benchmark problems and scale well to larger problems such as the handwritten digits example used in (Hinton et al., 1995). We believe there are two main reasons why the algorithm described here runs considerably faster than other Gibbs sampling based methods. The first is that there is no need to collect state statistics for each pattern. The weight values are reestimated using just one sampled internal state per pattern. The second is that weights that are not connected to informative units are not updated. This prevents the model from learning what are effectively overly strong priors and allows the weights in upper layers to adapt to structure actually in the data.

Gibbs sampling allows internal representations to be selected according to their true posterior probability. This was shown to be effective in cases where the resulting

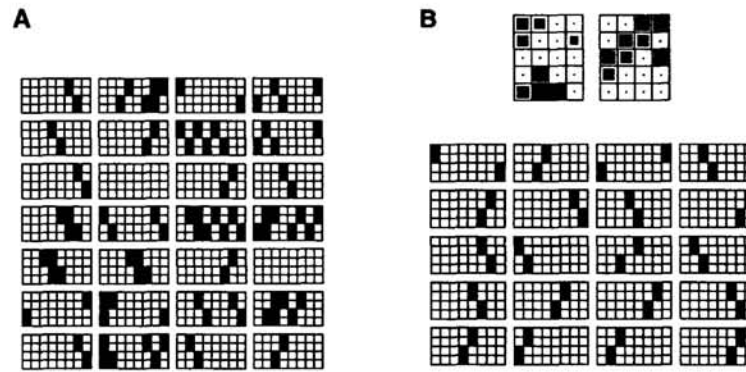

Figure 3: The shifter problem. (**A**) Input patterns are generated by generating a random binary vector in the bottom row. This pattern is shifted either left or right (with wrap around) with equal probability and copied to the top row. The input rows are duplicated to add redundancy as in Dayan et al. (1995). (**B**) The weights of a 32-20-2 after learning. The second layer of units learn to detect either local left shifts or right shifts in the data. These units cannot determine the shift direction alone, however, and require feedback from third layer units which integrate the outputs of all the units that represent a common shift (note that there is no overlap in the weights for the two third-layer units). This feedback turns off units that are inconsistent with the direction of shift. The weights that are close to zero for both third layer units effectively remove redundant second layer units that are not required to represent the input patterns.

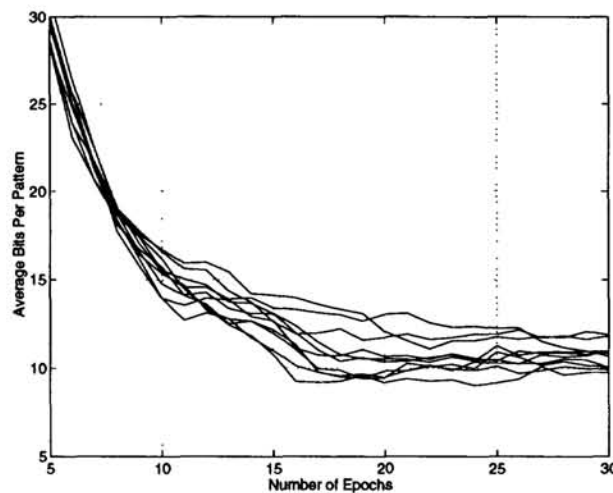

Figure 4: The graph shows 10 runs on the shifter problem from random initial conditions. The average bits per pattern is computed by $-\log(\mathcal{L})/(N\log 2)$. Each epoch used 200 input randomly generated input patterns. Two additional epochs were performed with 1000 random patterns to obtain accurate estimates of the average bits per pattern. The network converges rapidly and reliably. The best solutions, like the one shown in figure 3b, were found in 4/10 runs and had costs of approximately 10 bits at epoch 30. In this example, the network can get caught in local minima if too many units learn to represent the same local shifts.

representation has little ambiguity, *i.e.* each pattern has only a small number of probable explanations. If the causal structure to be learned is inherently ambiguous, *e.g.* in modeling the causal structure of medical symptoms, Gibbs sampling will be slow and better performance can be obtained with wake-sleep learning (Hinton et al., 1995; Frey et al., 1995) or mean field approximations (Saul et al., 1996).

There are many natural situations when there is ambiguity in low level features. This ambiguity can only be resolved by integrating the contextual information which itself is derived from the ambiguous simple features. This problem is common in the case of noisy input patterns and in feature grouping problems such as figure-ground separation. Feedback is crucial for ensuring that low-level representations are consistent within the larger context.

Some systems, such as the Helmholtz machine (Dayan et al., 1995; Hinton et al., 1995), arrive at the internal state through a feedforward process. It possible that this ambiguity in lower-level representations could be resolved by circuitry in the higher-level representations, but if multiple higher-level modules make use of the same low-level representations, the additional circuitry would have to be duplicated in each module. It seems more parsimonious to use feedback to influence the formation of the lower-level representations.

# References

Dayan, P., Hinton, G. E., Neal, R. M., and Zemel, R. S. (1995). The Helmholtz machine. *Neural Computation*, 7:889–904.

Földiák, P. (1989). Adaptive network for optimal linear feature extraction. In *Proceedings of the International Joint Conference on Neural Networks*, volume I, pages 401–405, Washington, D. C.

Frey, B. J., Hinton, G. E., and Dayan, P. (1995). Does the wake-sleep algorithm produce good density estimators? In Touretzky, D. S., Mozer, M., and Hasselmo, M., editors, *Advances in Neural Information Processing Systems*, volume 8, pages 661–667, San Mateo. Morgan Kaufmann.

Hinton, G. E., Dayan, P., Frey, B. J., and Neal, R. M. (1995). The wake-sleep algorithm for unsupervised neural networks. *Science*, 268(5214):1158–1161.

Hinton, G. E. and Sejnowski, T. J. (1986). Learning and relearning in Boltzmann machines. In Rumelhart, D. E. and McClelland, J. L., editors, *Parallel Distributed Processing*, volume 1, chapter 7, pages 282–317. MIT Press, Cambridge.

Lauritzen, S. L. and Spiegelhalter, D. J. (1988). Local computations with probabilities on graphical structures and their application to expert systems. *J. Royal Statistical Soc. Series B Methodological*, 50(2):157–224.

Neal, R. M. (1992). Connectionist learning of belief networks. *Artificial Intelligence*, 56(1):71–113.

Pearl, J. (1988). *Probabilistic Reasoning in Intelligent Systems*. Morgan Kaufmann, San Mateo.

Saul, L. K., Jaakkola, T., and Jordan, M. I. (1996). Mean field theory for sigmoid belief networks. *J. Artificial Intelligence Research*, 4:61–76.
